# Robust Reinforcement Learning in Motion Planning

**Satinder P. Singh***
Department of Brain and Cognitive Sciences
Massachusetts Institute of Technology
Cambridge, MA 02139
singh@psyche.mit.edu

**Andrew G. Barto, Roderic Grupen, and Christopher Connolly**
Department of Computer Science
University of Massachusetts
Amherst, MA 01003

## Abstract

While exploring to find better solutions, an agent performing on-line reinforcement learning (RL) can perform worse than is acceptable. In some cases, exploration might have unsafe, or even catastrophic, results, often modeled in terms of reaching 'failure' states of the agent's environment. This paper presents a method that uses domain knowledge to reduce the number of failures during exploration. This method formulates the set of actions from which the RL agent composes a control policy to ensure that exploration is conducted in a policy space that excludes most of the unacceptable policies. The resulting action set has a more abstract relationship to the task being solved than is common in many applications of RL. Although the cost of this added safety is that learning may result in a suboptimal solution, we argue that this is an appropriate tradeoff in many problems. We illustrate this method in the domain of motion planning.

An agent using reinforcement learning (Sutton et al., 1991; Barto et al., to appear) (RL) to approximate solutions to optimal control problems has to search, or explore, to improve its policy for selecting actions. Although exploration does not directly affect performance (Moore & Atkeson, 1993) in off-line learning with a model of the environment, exploration in on-line learning can lead the agent to perform worse than is acceptable. In some cases, exploration might have unsafe, or even catastrophic, results, often modeled in terms of reaching 'failure' states of the agent's environment. To make on-line RL more practical, especially if it involves expensive hardware, task-specific minimal levels of performance should be ensured during learning, a topic not addressed by prior RL research.

Although the need for exploration cannot be entirely removed, domain knowledge can sometimes be used to define the set of actions from which the RL agent composes a control policy so that exploration is conducted in a space that excludes most of the unacceptable policies. We illustrate this approach using a simulated dynamic mobile robot in two different environments.

## 1   Closed-loop policies as actions

RL agents search for optimal policies in a solution space determined in part by the set of actions available to the agent. With a few exceptions (e.g., Mahadevan & Connell, 1990; Singh, 1992), researchers have formulated RL tasks with actions that are *primitive* in the sense that they are low-level, are available in very state, are executed open-loop, and last a single time-step. We propose that this is an arbitrary, and self-imposed, restriction, and that in general the set of actions can have a much more abstract relationship to the problem being solved. Specifically, what are considered 'actions' by the RL algorithm can themselves be closed-loop control policies that meet important subgoals of the task being solved.

In this paper, the following general advantages afforded by using closed-loop policies as actions are demonstrated in the domain of motion planning:

1. It is possible to design actions to meet certain hard constraints so that RL maintains acceptable performance while simultaneously improving performance over time.

2. It is possible to design actions so that the action space for the learning problem has fewer dimensions than the actual dimension of the physical action space.

The robustness and greatly accelerated learning resulting from the above factors can more than offset the cost of designing the actions. However, care has to be taken in defining the action space to ensure that the resulting policy space contains at least one policy that is close to optimal.

## 2   RL with Dirichlet and Neumann control policies

The motion planning problem arises from the need to give an autonomous robot the ability to plan its own motion, i.e., to decide what actions to execute in order to achieve a task specified by initial and desired spatial arrangements of objects.

First consider geometric path planning, i.e., the problem of finding safe paths for a robot with no dynamical constraints in an environment with stationary obstacles. A safe path in our context is one that avoids all obstacles and terminates in a desired configuration. Connolly (1992) has developed a method that generates safe paths by solving Laplace's equation in configuration space with boundary conditions determined by obstacle and goal configurations (also see, Connolly & Grupen, 1993). Laplace's equation is the partial differential equation

$$\nabla^2 \phi \;=\; \sum_{i=1}^{n} \frac{\partial^2 \phi}{\partial x_i^2} = 0, \tag{1}$$

whose solution is a *harmonic* function, $\phi$, with no interior local minima. In practice, a finite difference approximation to Equation 1 is solved numerically via Gauss Sidel relaxation on a mesh over configuration space. Safe paths are generated by gradient descent on the resulting approximate harmonic function. In the general motion planning problem, we are interested in finding control policies that not only keep the robot safe but also extremize some performance criterion, e.g., minimum time, minimum jerk, etc.

Two types of boundary conditions, called Dirichlet and Neumann boundary conditions, give rise to two different harmonic functions, $\Phi_D$ and $\Phi_N$, that generate different types of safe paths. Robots following paths generated from $\Phi_D$ tend to be repelled perpendicularly from obstacles. In contrast, robots following paths generated from $\Phi_N$ tend to skirt obstacles by moving parallel to their boundaries. Although the state space in the motion planning problem for a dynamic robot in a planar environment is $\{x, \dot{x}, y, \dot{y}\}$, harmonic functions are derived in two-dimensional position space. These functions are inexpensive to compute (relative to the expense of solving the optimal control problem) because they are independent of the robot dynamics and criterion of optimal control. The closed-loop control policies that follow the gradient of the Dirichlet or Neumann solutions, respectively denoted $\pi_D$ and $\pi_N$, are defined approximately as follows: $\pi_D(s) = \nabla \Phi_D(\hat{s})$, and $\pi_N(s) = \nabla \Phi_N(\hat{s})$, where $\hat{s}$ is the projection of the state $s$ onto position space.[1]

Instead of formulating the motion planning problem as a RL task in which a control policy maps states into primitive control actions, consider the formulation in which a policy maps each state $s$ to a *mixing* parameter $k(s)$ so that the actual action is : $[1 - k(s)]\pi_D(s) + [k(s)]\pi_N(s)$, where $0 \le k(s) \le 1$. Figure 1B shows the structure of this kind of policy. In Appendix B, we present conditions guaranteeing that for a robot with no dynamical constraints, this policy space contains only acceptable policies, i.e., policies that cause the robot to reach the goal configuration without contacting an obstacle. Although this guarantee does not strictly hold when the robot has dynamical constraints, e.g., when there are bounds on acceleration, this formulation still reduces the risk of unacceptable behavior.

## 3  Simulation Results

In this paper we present a brief summary of simulation results for the two environments shown in Figures 2A and 3A. See Singh (1993) for details. The first

environment consists of two rooms connected by a corridor. The second environment is a horseshoe-shaped corridor. The mobile robot is simulated as a unit-mass that can accelerate in any direction. The only dynamical constraint is a bound on the maximum acceleration.

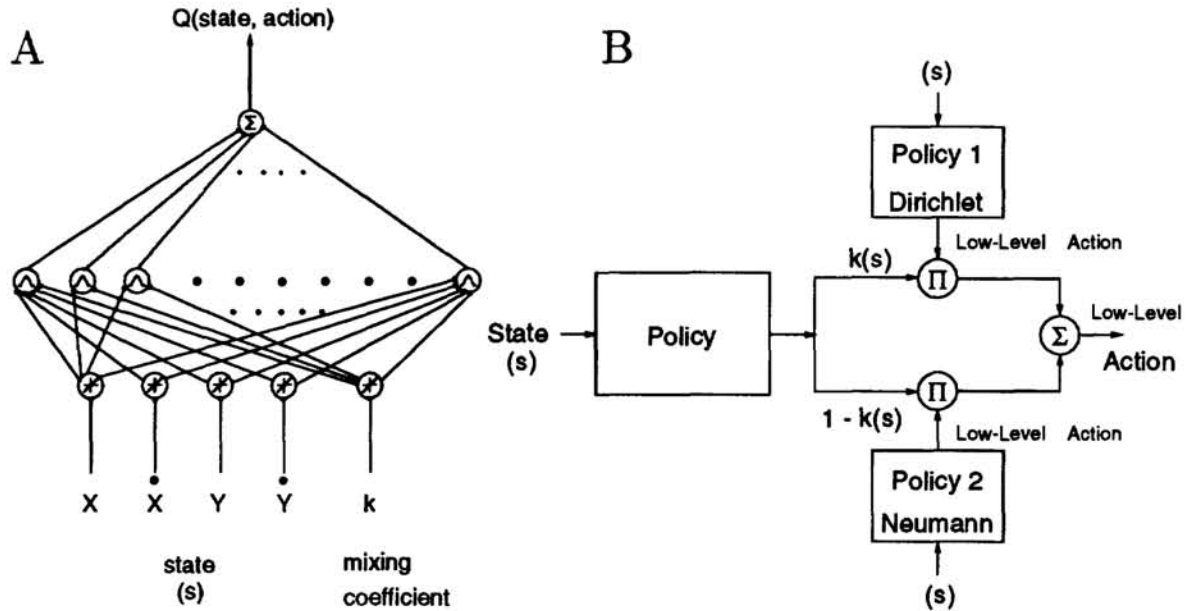

Figure 1: Q-learning Network and Policy Structure. Panel A: 2-layer Connectionist Network Used to Store Q-values. Network inversion was used to find the maximum Q-value (Equation 2) at any state and the associated greedy action. The hidden layer consists of radial-basis units. Panel B: Policy Structure. The agent has to learn a mapping from state $s$ to a mixing coefficient $0 \leq k(s) \leq 1$ that determines the proportion in which to mix the actions specifies by the pure Dirichlet and Neumann policies.

The learning task is to approximate *minimum time* paths from every point inside the environment to the goal region without contacting the boundary wall. A reinforcement learning algorithm called Q-learning (Watkins, 1989) (see Appendix A) was used to learn the mixing function, $k$. Figure 1A shows the 2-layer neural network architecture used to store the Q-values. The robot was trained in a series of trials, each trial starting with the robot placed at a randomly chosen state and ending when the robot enters the goal region. The points marked by stars in Figures 2A and 3A were the starting locations for which statistics were collected to produce learning curves.

Figures 2B, 2C, 3A and 3B show three robot trajectories from two randomly chosen start states; the black-filled circles mark the Dirichlet trajectory (labeled D), the white-filled circles mark the Neumann trajectory (labeled N), and the grey-filled circles mark the trajectory after learning (labeled Q). Trajectories are shown by taking snapshots of the robot at every time step; the velocity of the robot can be judged by the spacing between successive circles on the trajectory. Figure 2D shows the mixing function for zero-velocity states in the two-room environment, while Figure 3C shows the mixing function for zero velocity states in the horseshoe environment. The darker the region, the higher the proportion of the Neumann

policy in the mixture. In the two-room environment, the agent learns to follow the Neumann policy in the left-hand room and to follow the Dirichlet policy in the right-hand room.

Figure 2E shows the average time to reach the goal region as a function of the number of trials in the two-room environment. The solid-line curve shows the performance of the Q-learning algorithm. The horizontal lines show the average time to reach the goal region for the designated pure policies. Figure 3D similarly presents the results for the horseshoe environment. Note that in both cases the RL agent learns a policy that is better than either the pure Dirichlet or the pure Neumann policies. The relative advantage of the learned policy is greater in the two-room environment than in the horseshoe environment.

On the two-room environment we also compared Q-learning using harmonic functions, as described above, with Q-learning using primitive accelerations of the mobile robot as actions. The results are summarized along three dimensions: a) speed of learning: the latter system took more than $20,000$ trials to achieve the same level of performance achieved by the former in 100 trials, b) safety: the simulated robot using the latter system crashed into the walls more than 200 times, and c) asymptotic performance: the final solution found by the latter system was 6% better than the one found by the former.

## 4   Conclusion

Our simulations show how an RL system is capable of maintaining acceptable performance while simultaneously improving performance over time. A secondary motivation for this work was to correct the erroneous impression that the proper, if not the only, way to formulate RL problems is with low-level actions. Experience on large problems formulated in this fashion has contributed to the perception that RL algorithms are hopelessly slow for real-world applications. We suggest that a more appropriate way to apply RL is as a "component technology" that uses experience to improve on partial solutions that have already been found through either analytical techniques or the cumulative experience and intuitions of the researchers themselves. The RL framework is more abstract, and hence more flexible, than most current applications of RL would lead one to believe. Future applications of RL should more fully exploit the flexibility of the RL framework.

## A   Q-learning

On executing action $a$ in state $s_t$ at time $t$, the following update on the Q-value function is performed:

$$Q_{t+1}(s_t, a) = Q_t(s_t, a) + \alpha_t(s_t, a) \left[ R(s_t, a) + \gamma(\max_{a' \in A} Q_t(s_{t+1}, a')) - Q_t(s_t, a) \right] \quad (2)$$

where $R(s_t, a)$ is the payoff, $0 \le \gamma \le 1$ is the discount factor, and $\alpha$ is a learning rate parameter. See Watkins (1989) for further details.

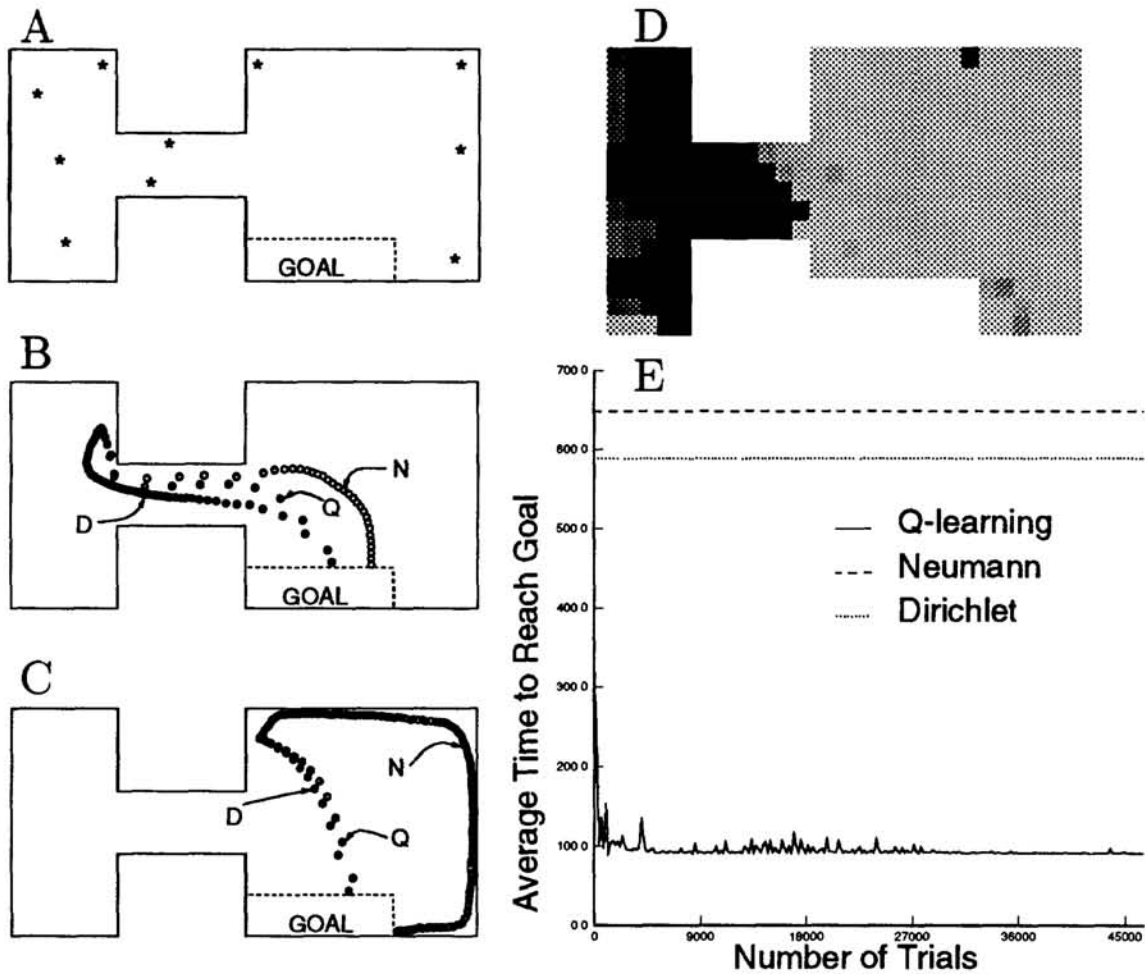

Figure 2: Results for the Two-Room Environment. Panel A: Two-Room Environment. The stars mark the starting locations for which statistics were computed. Panel B: Sample Trajectories from one Starting Location. The black-filled circles labeled D show a pure Dirichlet trajectory, the white-filled circles labeled N show a pure Neumann trajectory, and the grey-filled circles labeled Q show the trajectory after learning. The trajectories are shown by taking snapshots at every time step; the velocity of the robot can be judged by the distance between successive points on the trajectory. Panel C: Three Sample Trajectories from a Different Starting Location. Panel D: Mixing Function Learned by the Q-learning Network for Zero Velocity States. The darker the region the higher the proportion of the Neumann policy in the resulting mixture. Panel E: Learning Curve. The curve plots the time taken by the robot to reach the goal region, averaged over the locations marked with stars in Panel A, as a function of the number of Q-learning trials. The dashed line shows the average time using the pure Neumann policy; the dotted line for the pure Dirichlet policy; and the solid line for Q-learning. The mixed policy formed by Q-learning rapidly outperforms both pure harmonic function policies.

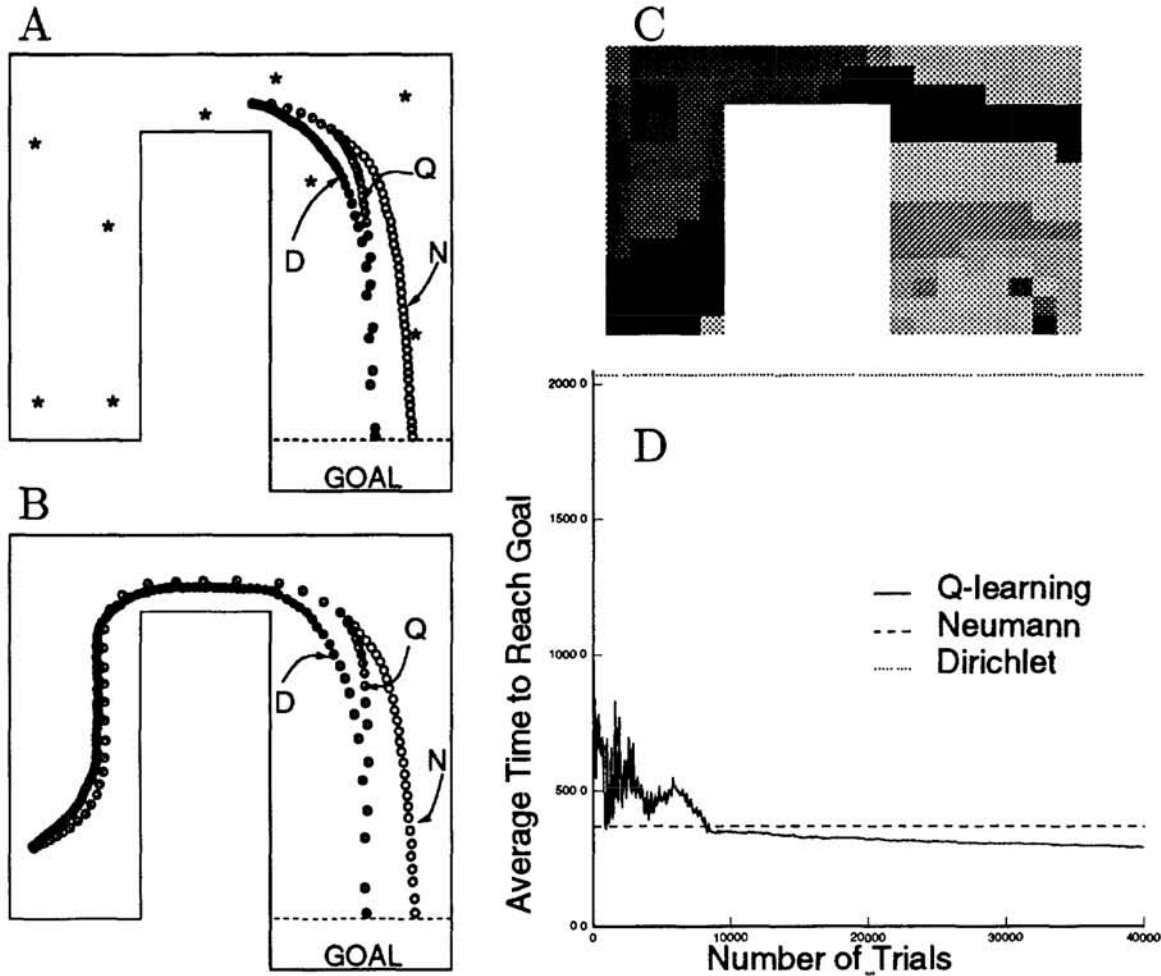

Figure 3: Results for the Horseshoe Environment. Panel A: Horseshoe-Shaped Environment. Locations marked with stars are the starting locations for which statistics were computed. It also shows sample trajectories from one starting location; the black-filled circles marked D show a Dirichlet trajectory, the white-filled circles marked N show a Neumann trajectory, and the grey-filled circles marked Q show the trajectory after learning. The trajectories are shown by taking snapshots at every time step; the velocity of the robot can be judged by the distance between successive points on the trajectory. Panel B: Three Sample Trajectories from a Different Starting Location. Panel C: Mixing Function Learned by the Q-learning Network for Zero Velocity States. The darker the region the higher the proportion of the Neumann policy in the resulting mixture. Panel D: Learning Curve. The curve plots the time taken by the robot to reach the goal region, averaged over the locations marked with stars in Panel A, as a function of the number of Q-learning trials. The dashed line shows the average time for the pure Neumann policy; the dotted line for the pure Dirichlet policy; and the solid line for Q-learning. Q-learning rapidly outperforms both pure harmonic function policies.

## B    Safety

Let $L$ denote the surface whose gradients at any point are given by the closed-loop policy under consideration. Then for there to be no minima in $L$, the gradient of $L$ should not vanish in the workspace, i.e., $(1 - k(s))\nabla\Phi_D(\hat{s}) + k(s)\nabla\Phi_N(\hat{s}) \neq 0$. The only way it can vanish is if $\forall i$

$$\frac{k(s)}{1 - k(s)} = \left[\frac{-\nabla\phi_D(\hat{s})}{\nabla\phi_N(\hat{s})}\right]_i, \qquad (3)$$

where $[\cdot]_i$ is the $i^{\text{th}}$ component of vector $[\cdot]$. The algorithm can explicitly check for that possibility and prevent it. Alternatively, note that due to the finite precision in any practical implementation, it is extremely unlikely that Equation 3 will hold in any state. Also note that $\pi(s)$ for any point $s$ on the boundary will always point away from the boundary because it is the convex sum of two vectors, one of which is normal to the boundary, and the other of which is parallel to the boundary.

### Acknowledgements

This work was supported by a grant ECS-9214866 to Prof. A. G. Barto from the National Science Foundation, and by grants IRI-9116297, IRI-9208920, and CDA-8922572 to Prof. R. Grupen from the National Science Foundation.

## Footnotes

*This work was done while the first author was finishing his Ph.D in computer science at the University of Massachusetts, Amherst.

[1]In practice, the gradients of the harmonic functions act as reference signals to a PD-controller.

## References

Barto, A.G., Bradtke, S.J., & Singh, S.P. (to appear). Learning to act using real-time dynamic programming. *Artificial Intelligence*.

Connolly, C. (1992). Applications of harmonic functions to robotics. In *The 1992 International Symposium on Intelligent Control*. IEEE.

Connolly, C. & Grupen, R. (1993). On the applications of harmonic functions to robotics. *Journal of Robotic Systems*, *10*(7), 931–946.

Mahadevan, S. & Connell, J. (1990). Automatic programming of behavior-based robots using reinforcement learning. Technical report, IBM Research Division, T.J.Watson Research Center, Yorktown Heights, NY.

Moore, A.W. & Atkeson, C.G. (1993). Prioritized sweeping: Reinforcement learning with less data and less real time. *Machine Learning*, *13*(1).

Singh, S.P. (1992). Transfer of learning by composing solutions for elemental sequential tasks. *Machine Learning*, *8*(3/4), 323–339.

Singh, S.P. (1993). *Learning to Solve Markovian Decision Processes*. PhD thesis, Department of Computer Science, University of Massachusetts. also, CMPSCI Technical Report 93-77.

Sutton, R.S., Barto, A.G., & Williams, R.J. (1991). Reinforcement learning is direct adaptive optimal control. In *Proceedings of the American Control Conference*, pages 2143–2146, Boston, MA.

Watkins, C.J.C.H. (1989). *Learning from Delayed Rewards*. PhD thesis, Cambridge Univ., Cambridge, England.
